# Bach in a Box - Real-Time Harmony

**Randall R. Spangler and Rodney M. Goodman***
Computation and Neural Systems
California Institute of Technology, 136-93
Pasadena, CA 91125

**Jim Hawkins**[†]
88B Milton Grove
Stoke Newington, London N16 8QY, UK

## Abstract

We describe a system for learning J. S. Bach's rules of musical harmony. These rules are learned from examples and are expressed as rule-based neural networks. The rules are then applied in *real-time* to generate new accompanying harmony for a live performer. Real-time functionality imposes constraints on the learning and harmonizing processes, including limitations on the types of information the system can use as input and the amount of processing the system can perform. We demonstrate algorithms for generating and refining musical rules from examples which meet these constraints. We describe a method for including *a priori* knowledge into the rules which yields significant performance gains. We then describe techniques for applying these rules to generate new music in real-time. We conclude the paper with an analysis of experimental results.

## 1 Introduction

The goal of this research is the development of a system to learn musical rules from examples of J.S. Bach's music, and then to apply those rules in *real-time* to generate new music in a similar style. These algorithms would take as input a melody such

[†]jhawkins@cix.compulink.co.uk

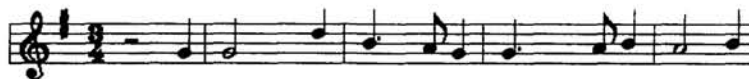

Figure 1: Melody for Chorale #1 "Aus meines Herzens Grunde"

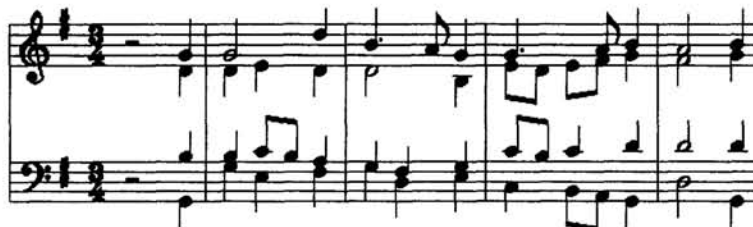

Figure 2: J. S. Bach's Harmony For Chorale #1

as Figure 1 and produce a complete harmony such as Figure 2. Performance of this harmonization in real-time is a challenging problem. It also provides insight into the nature of composing music.

We briefly review the representation of input data and the process of rulebase generation. Then we focus on methods of increasing the performance of rule-based systems. Finally we present our data on learning the style of Bach.

## 1.1 Constraints Imposed by Real-Time Functionality

A program which is to provide real-time harmony to accompany musicians at live performances faces two major constraints.

First, the algorithms must be fast enough to generate accompaniment without detectable delay between the musician playing the melody and the algorithm generating the corresponding harmony. For musical instrument sounds with sharp attacks (plucked and percussive instruments, such as the harp or piano), delays of even a few tens of milliseconds between the start of the melody note and the start of the harmony notes are noticeable and distracting. This limits the complexity of the algorithm and the amount of information it can process for each timestep.

Second, the algorithms must base their output only on information from previous timesteps. This differentiates our system from HARMONET (Hild, Feulnzer and Menzel, 1992) which required knowledge of the next note in the future before generating harmony for the current note.

## 1.2 Advantages of a Rule-Based Algorithm

A rule-based neural network algorithm was chosen over a recurrent network or a non-linear feed-forward network. Neural networks have been previously used for harmonizing music with some success (Mozer, 1991)(Todd, 1989). However, rule-based algorithms have several advantages when dealing with music. Almost all music has some sort of rhythm and is tonal, meaning both pitch and duration of individual notes are quantized. This presents problems in the use of continuous networks, which must be overtrained to reasonably approximate discrete behavior.

Rule-based systems are inherently discrete, and do not have this problem.

Furthermore it is very difficult to determine why a non-linear multi-layer network makes a given decision or to extract the knowledge contained in such a network. However, it is straightforward to determine why a rule-based network produced a given result by examining the rules which fired. This aids development of the algorithm, since it is easier to determine where mistakes are being made. It allows comparison of the results to existing knowledge of music theory as shown below, and may provide insight into the theory of musical composition beyond that currently available.

Rule-based neural networks can also be modified via segmentation to take advantage of additional *a priori* knowledge.

## 2 Background

### 2.1 Representation of Input Data

The choice of input representation greatly affects the ability of a learning algorithm to generate meaningful rules. The learning and inferencing algorithms presented here speak an extended form of the classical figured bass representation common in Bach's time. Paired with a melody, figured bass provides a sufficient amount of information to reconstruct the harmonic content of a piece of music.

Figured bass has several characteristics which make it well-disposed to learning rules. It is a symbolic format which uses a relatively small alphabet of symbols. It is also hierarchical - it specifies first the chord function that is to be played at the current note/timestep, then the scale step to be played by the bass voice, then additional information as needed to specify the alto and tenor scale steps. This allows our algorithm to fire sets of rules sequentially, to first determine the chord function which should be associated with a new melody note, and then to use that chord function as an input attribute to subsequent rulebases which determine the bass, alto, and tenor scale steps. In this way we can build up the final chord from simpler pieces, each governed by a specialized rulebase.

### 2.2 Generation of Rulebases

Our algorithm was trained on a set of 100 harmonized Bach chorales. These were translated from MIDI format into our figured bass format by a preprocessing program which segmented them into chords at points where any voice changed pitch. Chord function was determined by simple table lookup in a table of 120 common Bach chords based on the scale steps played by each voice in the chord. The algorithm was given information on the current timestep (Mel0-Te0), and the previous two timesteps (Mel1-Func2). This produced a set of 7630 training examples, a subset of which are shown below:

| Mel0 | Func0 | So0 | Ba0 | Al0 | Te0 | Mel1 | Func1 | So1 | Ba1 | Al1 | Te1 | Mel2 | Func2 |
|------|-------|-----|-----|-----|-----|------|-------|-----|-----|-----|-----|------|-------|
| D | V | S2 | B1 | A2 | T0 | E | I | S1 | B0 | A0 | T2 | C | I |
| E | I7 | S1 | B3 | A0 | T2 | D | V | S2 | B1 | A2 | T0 | E | I |
| F | IV | S0 | B1 | A2 | T1 | E | I7 | S1 | B3 | A0 | T2 | D | V |
| G | V | S0 | B0 | A1 | T2 | F | IV | S0 | B1 | A2 | T1 | E | I7 |

A rulebase is a collection of rules which predict the same right hand side (RHS) attribute (for example, Function0). All rules have the form **IF Y=y... THEN X=x.** A rule's order is the number of terms on its left hand side (LHS).

Rules are generated from examples using a modified version of the ITRULE algorithm. (Goodman et al., 1992) All possible rules are considered and ranked by a measure of the information contained in each rule defined as

$$ J(\mathbf{X}; \mathbf{Y} = y) = p(y) \left[ p(x|y) log \left( \frac{p(x|y)}{p(x)} \right) + (1 - p(x|y)) log \left( \frac{1 - p(x|y)}{1 - p(x)} \right) \right] \quad (1) $$

This measure trades off the amount of information a rule contains against the probability of being able to use the rule. Rules are less valuable if they contains little information. Thus, the J-measure is low when $p(x|y)$ is not much higher than $p(x)$. Rules are also less valuable if they fire only rarely ($p(y)$ is small) since those rules are unlikely to be useful in generalizing to new data.

A rulebase generated to predict the current chord's function might start with the following rules:

```
                                          p(corr)  J-meas
1. IF  Melody0     E   THEN Function0  I   0.621    0.095

2. IF  Function1   V   THEN Function0  V7  0.624    0.051
   AND Melody1     D
   AND Melody0     D

3. IF  Function1   V   THEN Function0  V7  0.662    0.049
   AND Melody0     D
```

## 2.3 Inferencing Using Rulebases

Rule based nets are a form of probabilistic graph model. When a rulebase is used to infer a value, each rule in the rulebase is checked in order of decreasing rule J-measure. A rule can fire if it has not been inhibited and all the clauses on its LHS are true. When a rule fires, its weight is added to the weight of the value which it predicts, After all rules have had a chance to fire, the result is an array of weights for all predicted values.

## 2.4 Process of Harmonizing a Melody

Input is received a note at a time as a musician plays a melody on a MIDI keyboard. The algorithm initially knows the current melody note and the data for the last two timesteps. The system first uses a rulebase to determine the chord function which should be played for the current melody note. For example, given the melody note "C", it might play a chord function "IV", corresponding to an F-Major chord. The program then uses additional rulebases to specify how the chord will be voiced. In the example, the bass, alto, and tenor notes might be set to "B0", "A1", and "T2", corresponding to the notes "F", "A", and "C". The harmony notes are then converted to MIDI data and sent to a synthesizer, which plays them in real-time to accompany the melody.

# 3 Improvement of Rulebases

The J-measure is a good measure for determining the information-theoretic worth of rules. However, it is unable to take into account any additional *a priori* knowledge about the nature of the problem - for example, that harmony rules which use the current melody note as input are more desirable because they avoid dissonance between the melody and harmony.

## 3.1 Segmentation

*A priori* knowledge of this nature is incorporated by segmenting rulebases into more- and less-desirable rules based on the presence or absence of a desired LHS attribute such as the current melody note (Melody0). Rules lacking the attribute are removed from the primary set of rules and placed in a second "fallback" set. Only in the event that no primary rules are able to fire is the secondary set allowed to fire. This gives greater impact to the primary rules (since they are used first) without the loss of domain size (since the less desirable rules are not actually deleted).

Rulebase segmentation provides substantial improvements in the speed of the algorithm in addition to improving its inferencing ability. When an unsegmented rulebase is fired, the algorithm has to compare the current input data with the LHS of every rule in the rulebase. However, processing for a segmented rulebase stops after the first segment which fires a rule on the input data. The algorithm does not need to spend time examining rules in lower-priority segments of that rulebase. This increase in efficiency allows segmented rulebases to contain more rules without impacting performance. The greater number of rules provides a richer and more robust knowledge base for generating harmony.

## 3.2 Realtime Dependency Pruning

When rules are used to infer a value, the rules weights are summed to generate probabilities. This requires that all rules which are allowed to fire must be independent of one another. Otherwise, one good rule could be overwhelmed by the combined weight of twenty mediocre but virtually identical rules. To prevent this problem, each segment of a rulebase is analyzed to determine which rules are dependent with other rules in the same segment. Two rules are considered dependent if they fire together on more than half the training examples where either rule fires.

For each rule, the algorithm maintains a list of lower rank rules which are dependent with the rule. This list is used in real-time dependency pruning. Whenever a rule fires on a given input, all rules dependent on it are inhibited for the duration of the input. This ensures that all rules which are able to fire for an input are independent.

## 3.3 Conflict Resolution

When multiple rules fire and predict different values, an algorithm must be used to resolve the conflict. Simply picking the value with the highest weight, while most likely to be correct, leads to monotonous music since a given melody then always produces the same harmony.

To provide a more varied harmony, our system exponentiates the accumulated rule

### Table 1: Rulebase Segments

| RHS | REQUIRED LHS FOR SEGMENT | RULES |
|---|---|---|
| Function0 | Melody0, Function1, Function2 | 110 |
| | Melody0, Function1 | 380 |
| | Melody0 | 346 |
| Soprano0 | Melody0, Function0 | 74 |
| Bass0 | Function0, Soprano0 | 125 |
| | (none) | 182 |
| Alto0 | Soprano0, Bass0 | 267 |
| | (none) | 533 |
| Tenor0 | Soprano0, Bass0, Alto0, Function0 | 52 |
| | Soprano0, Bass0, Alto0 | 164 |
| | (none) | 115 |

### Table 2: Rulebase Performance

| RHS | RULEBASE | RULES | AVG EVAL | CORRECT |
|---|---|---|---|---|
| Function0 | unsegmented | 1825 | 1825 | 55% |
| | segmented | 816 | 428 | 56% |
| | unsegmented #2 | 428 | 428 | 50% |
| Soprano0 | unsegmented | 74 | 74 | 95% |
| Bass0 | unsegmented | 307 | 307 | 70% |
| | segmented | 307 | 162 | 70% |
| | unsegmented #2 | 162 | 162 | 65% |
| Alto0 | unsegmented | 800 | 800 | 63% |
| | segmented | 800 | 275 | 63% |
| | unsegmented #2 | 275 | 275 | 59% |
| Tenor0 | unsegmented | 331 | 331 | 73% |
| | segmented | 331 | 180 | 74% |
| | unsegmented #2 | 180 | 180 | 67% |

weights for the possible outcomes to produce probabilities for each value, and the final outcome is chosen randomly based on those probabilities. It is because we use the accumulated rule weights to determine these probabilities that all rules which are allowed to fire must be independent of each other.

If no rules at all fire, the system uses a first-order Bayes classifier to determine the RHS value based on the current melody note. This ensures that the system will always return an outcome compatible with the melody.

## 4   Results

Rulebases were generated for each attribute. Up to 2048 rules were kept in each rulebase. Rules were retained if they were correct at least 30% of the time they fired, and had a J-measure greater than 0.001. The rulebases were then segmented.

These rulebases were tested on 742 examples derived from 27 chorales not used in the training set. The number of examples correctly inferenced is shown for each rulebase before and after segmentation. Also shown is the average number of rules evaluated per test example; the speed of inferencing is proportional to this number.

To determine whether segmentation was in effect only removing lower J-measure rules, we removed low-order rules from the unsegmented rulebases until they had the same average number of rules evaluated as the segmented rulebases.

In all cases, segmenting the rulebases reduced the average rules fired per example without lowering the accuracy of the rulebases (in some cases, segmentation even increased accuracy). Speed gains from segmentation ranged from 80% for Tenor0 up to 320% for Function0. In comparison, simply reducing the size of the unsegmented

rulebase to match the speed of the segmented rulebase reduced the number of correctly inferred examples by 4% to 6%.

The generated rules for harmony have a great deal of similarity to accepted harmonic transitions (Ottman, 1989). For example, high-priority rules specify common chord transitions such as V-V7-I (a classic way to end a piece of music).

## 5 Remarks

The system described in this paper meets the basic objectives described in Section 1. It learns harmony rules from examples of the music of J.S. Bach. The system is then able to harmonize melodies in real-time. The generated harmonies are sometimes surprising (such as the diminished 7th chord near the end of "Happy Birthday"), yet are consistent with Bach harmony.

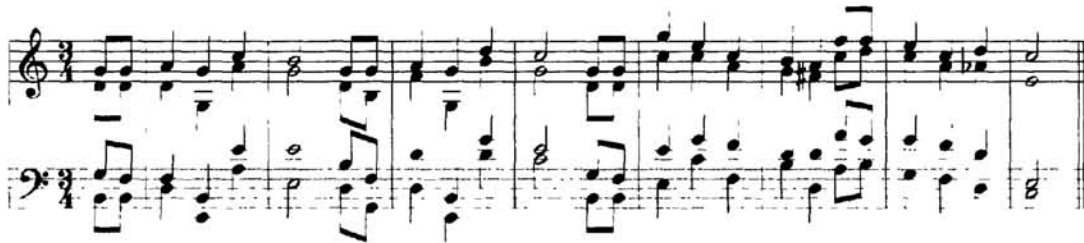

Figure 3: Algorithm's Bach-Like Harmony for "Happy Birthday"

Rulebase segmentation is an effective method for incorporating *a priori* knowledge into learned rulebases. It can provides significant speed increases over unsegmented rulebases with no loss of accuracy.

### Acknowledgements

Randall R. Spangler is supported in part by an NSF fellowship.

## Footnotes

*rspangle@micro.caltech.edu, rogo@micro.caltech.edu

### References

J. Bach (Ed.: A. Riemenschneider) (1941) 371 Harmonized Chorales and 96 Chorale Melodies. Milwaukee, WI: G. Schirmer.

H. Hild, J. Feulner & W. Menzel. (1992) HARMONET: A Neural Net for Harmonizing Chorales in the Style of J. S. Bach. In J. Moody (ed.), *Advances in Neural Information Processing Systems 4*, 267-274. San Mateo, CA: Morgan Kaufmann.

M. Mozer, T. Soukup. (1991) Connectionist Music Composition Based on Melodic and Stylistic Constraints. In R. Lippmann (ed.), *Advances in Neural Information Processing Systems 3*. San Mateo, CA: Morgan Kaufmann.

P. Todd. (1989) A Connectionist Approach to Algorithmic Composition. *Computer Music Journal* **13**(4):27-43.

R. Goodman, P. Smyth, C. Higgins, J. Miller. (1992) Rule-Based Neural Networks for Classification and Probability Estimation. *Neural Computation* 4(6):781-804.

R. Ottman. (1989) Elementary Harmony. Englewood Cliffs, NJ: Prentice Hall.